# Reinforcement Learning to Play an Optimal Nash Equilibrium in Team Markov Games

**Xiaofeng Wang**
ECE Department
Carnegie Mellon University
Pittsburgh, PA 15213
xiaofeng@andrew.cmu.edu

**Tuomas Sandholm**
CS Department
Carnegie Mellon University
Pittsburgh, PA 15213
sandholm@cs.cmu.edu

## Abstract

Multiagent learning is a key problem in AI. In the presence of multiple Nash equilibria, even agents with non-conflicting interests may not be able to learn an optimal coordination policy. The problem is exaccerbated if the agents do not know the game and independently receive noisy payoffs. So, multiagent reinforfcement learning involves two interrelated problems: identifying the game and learning to play. In this paper, we present *optimal adaptive learning*, the first algorithm that converges to an optimal Nash equilibrium with probability 1 in any team Markov game. We provide a convergence proof, and show that the algorithm's parameters are easy to set to meet the convergence conditions.

## 1 Introduction

Multiagent learning is a key problem in AI. For a decade, computer scientists have worked on extending reinforcement learning (RL) to multiagent settings [11, 15, 5, 17]. Markov games (aka. stochastic games) [16] have emerged as the prevalent model of multiagent RL. An approach called *Nash-Q* [9, 6, 8] has been proposed for learning the game structure and the agents' strategies (to a fixed point called *Nash equilibrium* where no agent can improve its expected payoff by deviating to a different strategy). Nash-Q converges if a unique Nash equilibrium exists, but generally there are multiple Nash equilibria. Even team Markov games (where the agents have common interests) can have multiple Nash equilibria, only some of which are optimal (that is, maximize sum of the agents' discounted payoffs). Therefore, learning in this setting is highly nontrivial.

A straightforward solution to this problem is to enforce *convention* (social law). Boutilier proposed a tie-breaking scheme where agents choose individual actions in lexicographic order[1]. However, there are many settings where the designer is unable or unwilling to impose a convention. In these cases, agents need to learn to coordinate. Claus and Boutilier introduced *fictitious play*, an equilibrium selection technique in game theory, to RL. Their algorithm, *joint action learner* (JAL) [2], guarantees the convergence to a Nash equilibrium in a team stage game. However, this equilibrium may not be optimal. The same problem prevails in other equilibrium-selection approaches in game theory such as *adaptive play* [18] and the evolutionary model proposed in [7].

In RL, the agents usually do not know the environmental model (game) up front and receive noisy payoffs. In this case, even the lexicographic approaches may not work because agents receive noisy payoffs independently and thus may never perceive a tie. Another significant

problem in previous research is how a nonstationary exploration policy (required by RL) affects the convergence of equilibrium selection approaches—which have been studied under the assumption that agents either always take the best-response actions or make mistakes at a constant rate. In RL, learning to play an optimal Nash equilibrium in team Markov games has been posed as one of the important open problems [9]. While there have been heuristic approaches to this problm, no existing algorithm has been proposed that is guarenteed to converge to an optimal Nash equilibrium in this setting.

In this paper, we present *optimal adaptive learning* (OAL), the first algorithm that converge to an optimal Nash equilibrium with probability 1 in any team Markov game (Section 3). We prove its convergence, and show that OAL's parameters are easy to meet the convergence conditions (Section 4).

## 2 The setting

### 2.1 MDPs and reinforcement learning (RL)

In a Markov decision problem, there is one agent in the environment. A fully observable Markov decision problem (MDP) is a tuple $\langle S, A, R, T \rangle$ where $S$ is a finite state space; $A$ is the space of actions the agent can take; $R : S \times A \to \Re$ is a payoff function ($R(s, a)$ is the expected payoff for taking action $a$ in state $s$); and $T : S \times A \times S \to [0, 1]$ is a transition function ($T(s, a, s')$ is the probability of ending in state $s'$, given that action $a$ is taken in state $s$). An agent's deterministic *policy* (aka. *strategy*) is a mapping from states to actions. We denote by $\pi(s)$ the action that policy $\pi$ prescribes in state $s$. The objective is to find a policy $\pi$ that maximizes $\sum_{t=0}^{\infty} \gamma^t E(r_t | \pi)$, where $r_t$ is the payoff at time $t$, and $\gamma \in [0, 1)$ is a discount factor. There exists a deterministic optimal policy $\pi^*$ [12]. The Q-function for this policy, $Q^*$, is defined by the set of equations $Q^*(s, a) = R(s, a) + \gamma \sum_{s' \in S} T(s, a, s') \max_{a' \in A} Q^*(s', a')$. At any state $s$, the optimal policy chooses $\arg \max_a Q^*(s, a)$ [10].

Reinforcement learning can be viewed as a sampling method for estimating $Q^*$ when the payoff function $R$ and/or transition function $T$ are unknown. $Q^*(s, a)$ can be approximated by a function $Q_t(s, a)$ calculated from the agent's experience up to time $t$. The model-based approach uses samples to generate models of $R$ and $T$, and then iteratively computes $Q_t(s, a) = R_t(s, a) + \gamma \sum_{s' \in S} T_t(s, a, s') \max_{a' \in A} Q_{t-1}(s', a')$.

Based on $Q_t$, a *learning policy* assigns probabilities to actions at each state. If the learning policy has the "*Greedy in the Limit with Infinite Exploration*" (GLIE) property, then $Q_t$ will converge to $Q^*$ (with either a model-based or model-free approach) and the agent will converge in behavior to an optimal policy [14]. Using GLIE, every state-action pair is visited infinitely often, and in the limit the action selection is greedy with respect to the Q-function w.p.1. One common GLIE policy is Boltzmann exploration [14].

### 2.2 Multiagent RL in team Markov games when the game is unknown

A natural extension of an MDP to multiagent environments is a *Markov game* (aka. *stochastic game*) [16]. In this paper we focus on *team Markov games*, that are Markov games where each agent receives the same expected payoff (*in the presence of noise, different agent may still receive different payoffs at a particular moment.*). In other words, there are no conflicts between the agents, but learning the game structure and learning to coordinate are nevertheless highly nontrivial.

**Definition 1** *A team Markov game (aka identical-interest stochastic game) $\Gamma$ is a tuple $\langle \alpha, S, A, R, T \rangle$, where $\alpha$ is a set of n agents; $S$ is a finite state space; $A = \times A_{i=1...n}$ is a joint action space of n agents; $R : S \times A \to \Re$ is the common expected payoff function; and $T : S \times A \times S \to [0, 1]$ is a transition function.*

The objective of the $n$ agents is to find a deterministic *joint policy (aka. joint strategy aka. strategy profile)* $\pi = \{\pi_{i=1...n}\}$ (where $\pi : S \to A$ and $\pi_i : S \to A_i$) so as to maximize the

expected sum of their discounted payoffs. The Q-function, $Q^\pi(s, a)$, is the expected sum of discounted payoffs given that the agents play joint action $a$ in state $s$ and follow policy $\pi$ thereafter. The optimal Q-function $Q^*(s, a)$ is the Q-function for (each) optimal policy $\pi^*$. So, $Q^*$ captures the game structure. The agents generally do not know $Q^*$ in advance. Sometimes, they know neither the payoff structure nor the transition probabilities.

A joint policy $\{\pi_{i=1...n}\}$ is a *Nash equilibrium*[1] if each individual policy is a best response to the others. That is, for all $i \in \alpha$, $s \in S$ and any individual policy $\pi_i'$, $Q^*(s, \{\pi_i(s)\} \cup \pi_{-i}(s)) \geq Q^*(s, \{\pi_i'(s)\} \cup \pi_{-i}(s))$, where $\pi_{-i}$ is the joint policy of all agents except agent $i$. (Likewise, throughout the paper, we use $-i$ to denote all agents but $i$, e.g., by $a_{-i}$ their joint action, by $A_{-i}$ their joint action set.) A Nash equilibrium is *strict* if the inequality above is strict. An *optimal Nash equilibrium* $\pi^*$ is a Nash equilibrium that gives the agents the maximal expected sum of discounted payoffs. In team games, each optimal Nash equilibrium is an optimal joint policy (and there are no other optimal joint policies).

A joint action $a$ is optimal in state $s$ if $Q^*(s, a) \geq Q^*(s, a')$ for all $a' \in A$. If we treat $Q^*(s, a)$ as the payoff of joint action $a$ in state $s$, we obtain a team game in matrix form. We call such a game a *state game* for $s$. An optimal joint action in $s$ is an optimal Nash equilibrium of that state game. Thus, the task of optimal coordination in a team Markov game boils down to having all the agents play an optimal Nash equilibrium in state games.

However, a *coordination problem* arises if there are multiple Nash equilibria. *The 3-player*

| | $b1c1$ | $b1c2$ | $b1c3$ | $b2c1$ | $b2c2$ | $b2c3$ | $b3c1$ | $b3c2$ | $b3c3$ |
|---|---|---|---|---|---|---|---|---|---|
| $a1$ | 10 | -20 | -20 | -20 | -20 | 5 | -20 | 5 | -20 |
| $a2$ | -20 | -20 | 5 | -20 | 10 | -20 | 5 | -20 | -20 |
| $a3$ | -20 | 5 | -20 | 5 | -20 | -20 | -20 | -20 | 10 |

Table 1: *A three-player coordination game*

*game in Table 1 has three optimal Nash equilibria and six sub-optimal Nash equilibria*. In this game, no existing equilibrium selection algorithm (e.g.,*fictitious play* [3]) is guarenteed to learn to play an optimal Nash equilibrium. Furthermore, if the payoffs are only expectations over each agent's noisy payoffs and unknown to the agents before playing, even identification of these sub-optimal Nash equilibria during learning is nontrivial.

## 3 Optimal adaptive learning (OAL) algorithm

We first consider the case where agents know the game before playing. This enables the learning agents to construct a *virtual game* (VG) for each state $s$ of the team Markov game to eliminate all the strict suboptimal Nash equilibria in that state. Let $VG^*(s, a)$ be the payoff that the agents receive from the VG in state $s$ for a joint action $a$. We let $VG^*(s, a) = 1$ if $a = \arg\max_{a' \in A} Q^*(s, a')$ and $VG^*(s, a) = 0$ otherwise. For example, the VG for the game in Table 1 gives payoff 1 for each optimal Nash equilibrium $(\{a1, b1, c1\}, \{a2, b2, c2\}, \text{ and } \{a3, b3, c3\})$, and payoff 0 to every other joint action. The VG in this example is *weakly acyclic*.

**Definition 2 (Weakly acyclic game [18])** *Let $G$ be an n-player game in matrix form. The best-response graph of $G$ takes each joint action $a \in A$ as a vertex and connects two vertices $a$ and $a'$ with a directed edge $a \to a'$ if and only if 1) $a \neq a'$; 2) there exists exactly one agent $i$ such that $a_i'$ is a best response to $a_{-i}$ and $a_{-i}' = a_{-i}$. We say the game $G$ is weakly acyclic if in its best-response graph, from any initial vertex $a$, there exists a directed path to some vertex $a^*$ from which there is no outgoing edge.*

To tackle the equilibrium selection problem for weakly acyclic games, Young [18] proposed a learning algorithm called *adaptive play* (AP), which works as follows. Let $a^t \in A$ be a joint action played at time $t$ over an n-player game in matrix form. Fix integers

$k$ and $m$ such that $1 \leq k \leq m$. When $t \leq m$, each agent $i$ randomly chooses its actions. Starting from $t = m + 1$, each agent looks back at the $m$ most recent plays $h_t = (a^{t-m}, a^{t-m+1}, \ldots, a^{t-1})$ and randomly (without replacement) selects $k$ samples from $h_t$. Let $K_{t+1}(a_{-i})$ be the number of times that a reduced joint action $a_{-i} \in A_{-i}$ (a joint action without agent $i$'s individual action) appears in the $k$ samples at $t + 1$. Let $u_i(a)$ be agent $i$'s payoff given that joint action $a$ has been played. Agent $i$ calculates its expected payoff w.r.t its individual action $a_i$ as $EP(a_i) = \sum_{a_{-i} \in A_{-i}} u_i(\{a_i\} \cup a_{-i}) \frac{K_{t+1}(a_{-i})}{k}$, and then randomly chooses an action from a set of best responses: $BR_i^t = \{a_i \mid a_i = \arg\max_{a_i' \in A_i} EP(a_i')\}$.

Young showed that AP in a weakly acyclic game converges to a strict Nash equilibrium w.p.1. Thus, AP on the VG for the game in Table 1 leads to an equilibrium with payoff 1 which is actually an optimal Nash equilibrium for the original game. Unfortunately, this does not extend to all VGs because not all VGs are weakly acyclic: in a VG without any strict Nash equilibrium, AP may not converge to the strategy profile with payoff 1.

In order to address more general settings, we now modify the notion of weakly acyclic game and adaptive play to accommodate weak optimal Nash equilibria.

**Definition 3 (Weakly acyclic game w.r.t a biased set (WAGB))**: *Let $D$ be a set containing some of the Nash equilibria of a game $G$ (and no other joint policies). Game $G$ is a WAGB if, from any initial vertex $a$, there exists a directed path to either a Nash equilibrium inside $D$ or a strict Nash equilibrium.*

We can convert any VG to a WAGB by setting the biased set $D$ to include all joint policies that give payoff 1 (and no other joint policies). To solve such a game, we introduce a new learning algorithm for equilibrium selection. It enables each agent to deterministically select a best-response action once any Nash equilibrium in the biased set is attained (even if there exist several best responses when the Nash equilibrium is not strict). This is different from AP where players randomize their action selection when there are multiple best-response actions. We call our approach *biased adaptive play* (BAP). BAP works as follows. Let $D$ be the biased set composed of some Nash equilibria of a game in matrix form. Let $SP_t$ be the set of $k$ samples drawn at time $t$, without replacement, from among the most recent $m$ joint actions. If (1) there exists a joint action $a' \in D$ such that for all $a \in SP_t$, $a_{-i} \subset a$ and $a_{-i} \subset a'$, and (2) there exists at least one joint action $a$ such that $a \in SP_t$ and $a \in D$, then agent $i$ chooses its best-response action $a_i$ such that $a_i \in a^{t'}$ and $t' = \max\{T \mid a^T \in SP_t \wedge a^T \in D\}$. That is, $a_i$ is contained in the most recent play of a Nash equilibrium inside $D$. On the other hand, if the two conditions above are not met, then agent $i$ chooses its best-response action in the same way as AP. As we will show, BAP (even with GLIE exploration) on a WAGB converges w.p.1 to either a Nash equilibrium in $D$ or a strict Nash equilibrium.

So far we tackled learning of coordination in team Markov games where the game structure is known. Our real interests are in learning when the game is unknown. In multiagent reinforcement learning, $Q^*$ is asymptotically approximated with $Q_t$. Let $VG_t(s, a)$ be the virtual game w.r.t $Q_t(s, a)$. Our question is how to construct $VG_t$ so as to assure $VG_t \to VG^*$ w.p.1. Our method of achieving this makes use of the notion of $\varepsilon$-optimality.

**Definition 4** *Let $\varepsilon$ be a positive constant. A joint action $a$ is $\varepsilon$-optimal at state $s$ and time $t$ if $Q_t(s, a) + \varepsilon \geq \max_{a'} Q_t(s, a')$ for all $a' \in A$. We denote the set of $\varepsilon$-optimal joint actions at state $s$ and time $t$ as $A^{\varepsilon_t}(s)$.*

The idea is to use a decreasing $\varepsilon$-bound $\varepsilon_t$ to estimate $A^{\varepsilon_t}(s)$ at state $s$ and time $t$. All the joint actions belonging to the set are treated as optimal Nash equilibria in the virtual game $VG_t$ which give agents payoff 1. If $\varepsilon_t$ converges to zero at a rate slower than $Q_t$ converges

to $Q^*$, then $VG_t \to VG^*$ w.p.1. We make $\varepsilon_t$ proportional to a function $B(N_t) \in [0,1]$ which decreases slowly and monotonically to zero with $N_t$, where $N_t$ is the smallest number of times that any state-action pair has been sampled so far. Now, we are ready to present the entire *optimal adaptive learning* (OAL) algorithm. As we will present thereafter, we craft $B(N_t)$ carefully using an understanding of the convergence rate of a model-based RL algorithm that is used to learn the game structure.

**Optimal adaptive learning algorithm** (for agent $i$)

1. **Initialization** $t = 0$. For all $s \in S$ and $a \in A$ do $n_t(s,a) = 1$, $T_t(s,a,s') = \frac{1}{|s|}$ and $R_t(s,a) = 0$. $\varepsilon_t = C$; $A^{\varepsilon_t}(s) = A$. $D = A$.

2. **Learning of coordination policy** If $t \le m$, randomly select an action, otherwise do

   (a) Update the virtual game $VG_t$ at state $s$: $VG_t(s,a) = 1$ if $a \in A^{\varepsilon_t}(s)$ and $VG_t(s,a) = 0$ otherwise. Set $D = \{a \mid VG_t(s,a) = 1\}$.

   (b) According to GLIE exploration, with an exploitation probability $p$ do

      i. Randomly select (without replacement) $k$ records from $m$ recent observations of others' joint actions played at state $s$.

      ii. Calculate expected payoff of individual action $a_i$ over the virtual game $VG_t(s,a)$ at current state $s$ as follows: $EP(s,a_i) = \sum_{a_{-i} \in A_{-i}} \frac{K_t(s,a_{-i})}{k} VG_t(s, \{a_i\} \cup a_{-i})$. Construct the best-response set at state $s$ and time $t$: $BR_i^t(s) = \{a_i \mid a_i = \arg\max_{a_i' \in A_i} EP(s,a_i')\}$.

      iii. If conditions 1) and 2) of BAP are met, choose a best-response action with respect to the biased set $D$. Otherwise, randomly select a best-response action from $BR_i^t(s)$.

      Otherwise, randomly select an action to explore.

3. **Off-policy learning of game structure** $Q^*$

   (a) Observe state transition $s \to s'$ and payoff $r_t$ under the joint action $a$. Do

      i. $n_t(s,a) \leftarrow n_t(s,a) + 1$.

      ii. $R_t(s,a) \leftarrow R_t(s,a) + \frac{1}{n_t(s,a)}(r_t - R_t(s,a))$.

      iii. $T_t(s,a,s') \leftarrow T_t(s,a,s') + \frac{1}{n_t(s,a)}(1 - T_t(s,a,s'))$.

      iv. For all $s" \in S$ and $s" \ne s'$ do $T_t(s,a,s") \leftarrow (1 - \frac{1}{n_t(s,a)})T_t(s,a,s")$.

   (b) $Q_{t+1}(s,a) \leftarrow R_t(s,a) + \gamma \sum_{s' \in S} T_t(s,a,s') \max_{a' \in A} Q_t(s',a')$.

   (c) $t \leftarrow t + 1$. $N_t \leftarrow \min_{s \in S, a \in A} n_t(s,a)$.

   (d) If $\varepsilon_t > CB(N_t)$ (see Section 4.2 for the construction of $B(N_t)$)

      i. $\varepsilon_t \leftarrow CB(N_t)$.

      ii. Update $Q_t(s,a)$ for all $(s,a)$ using (b).

      iii. $A^{\varepsilon_t}(s) \leftarrow \{a \mid Q_t(s,a) + \varepsilon_t \ge \max_{a' \in A} Q_t(s,a')\}$.

Here, $n_t(s,a)$ is the number of times a joint action $a$ has been played in state $s$ by time $t$. $C$ is a positive constant (any value works). $K_t(s,a_{-i})$ is the number of times that a joint action $a_{-i} \in A_{-i}$ appears in agent $i$'s $k$ samples (at time $t$) from the most recent $m$ joint actions taken in state $s$.

# 4 Proof of convergence of OAL

In this section, we prove that OAL converges to an optimal Nash equilibrium. Throughout, we make the common RL assumptions: payoffs are bounded, and the number of states and actions is finite. The proof is organized as follows. In Section 4.1 we show that OAL agents learn optimal coordination if the game is known. Specifically, we show that BAP against a WAGB with known game structure converges to a Nash equilibrium under GLIE exploration. Then in Section 4.2 we show that OAL agents will learn the game structure. Specifically, any virtual game can be converted to a WAGB which will be learned surely. Finally, these two tracks merge in Section 4.3 which shows that OAL agents will learn the game structure and optimal coordination. Due to limited space, we omit most proofs. They can be found at: www.cs.cmu.edu/~sandholm/oal.ps.

## 4.1 Learning to coordinate in a known game

In this section, we first model our biased adaptive play (BAP) algorithm with best-response action selection as a stationary Markov chain. In the second half of this section we then model BAP with GLIE exploration as a nonstationary Markov chain.

### 4.1.1 BAP as a stationary Markov chain

Consider BAP with randomly selected initial $m$ plays. We take the initial $history$ $h_m = (a^1, a^2, \ldots, a^m)$ as the initial state of the Markov chain. The definition of the other states is inductive: A successor of state $h$ is any state $h'$ obtained by deleting the left-most element of $h$ and appending a new right-most element. The only exception is that all the states $h = (a, a, ..., a)$ with $a$ being either a member of the biased set $D$ or a strict Nash equilibrium are grouped into a unique terminal state $h_T$. Any state directing to $h \in h_T$ is treated as directly connected to $h_T$.

Let $P$ be the state transition matrix of the above Markov chain. Let $h'$ be a successor of $h$, and let $a = \{a_1, a_2, \ldots, a_n\}$ ($n$ players) be the new element that was appended to the right of $h$ to get $h'$. Let $p_{h,h'} > 0$ be the transition probability from $h$ to $h'$. Now, $p_{h,h'} > 0$ if and only if for each agent $i$, there exists a sample of size $k$ in $h$ to which $a_i$ is $i$'s best response according to the action-selection rule of BAP. Because agent $i$ chooses such a sample with a probability independent of time $t$, the Markov chain is stationary. Finally, due to our clustering of multiple states into a terminal state $h_T$, for any state $h$ connected to $h_T$, we have $p_{h,h_T} = \sum_{h' \in h_T} p_{h,h'}$.

In the above model, once the system reaches the terminal state, each agent's best response is to repeat its most recent action. This is straightforward if in the actual terminal state $h = (a, a, ..., a)$ (which is one of the states that were clustered to form the terminal state), $a$ is a strict Nash equilibrium. If $a$ is only a weak Nash equilibrium (in this case, $a \in D$), BAP biases each agent to choose its most recent action because conditions (1) and (2) of BAP are satisfied. Therefore, the terminal state $h_T$ is an absorbing state of the finite Markov chain.[2] On the other hand, the above analysis shows that $h_T$ essentially is composed of multiple absorbing states. Therefore, if agents come into $h_T$, they will be stuck in a particular state in $h_T$ forever instead of cycling around multiple states in $h_T$.

**Theorem 1** *Let G be a weakly acyclic game w.r.t. a biased set D. Let L(a) be the length of the shortest directed path in the best-response graph of G from a joint action a to either an absorbing vertex or a vertex in D, and let $L_G = \max_a L(a)$. If $m \geq k(L_G + 2)$, then, w.p.1, biased adaptive play in G converges to either a strict Nash equilibrium or a Nash equilibrium in D.*

Theorem 1 says that the stationary Markov chain for BAP in a WAGB (given $m \geq k(L_G + 2)$) has a unique stationary distribution in which only the terminal state appears.

### 4.1.2 BAP with GLIE exploration as a nonstationary Markov chain

Without knowing game structure, the learners need to use exploration to estimate their payoffs. In this section we show that such exploration does not hurt the convergence of BAP. We show this by first modeling BAP with GLIE exploration as a *non-stationary Markov chain*.

With GLIE exploration, at every time step $t$, each *joint action* occurs with positive probability. This means that the system transitions from the state it is in to any of the successor states with positive probability. On the other hand, the agents' action-selection becomes increasingly greedy over time. In the limit, with probability one, the transition probabilities converge to those of BAP with no exploration. Therefore, we can model the learning process with a sequence of transition matrices $\{P_t\}_{t=1}^{\infty}$ such that $\lim_{t \to \infty} P_t = P$, where $P$ is the transition matrix of the stationary Markov chain describing BAP without exploration.

Our objective here is to show that on a WAGB, BAP with GLIE exploration will converge to the ("clustered") terminal state. For that, we use the following lemma (which is a combination of Theorems V4.4 and V4.5 from [4]).

**Lemma 2** *Let $P$ be the finite transition matrix of a stationary Markov chain with a unique stationary distribution $g$. Let $\{P_t\}_{t=1}^{\infty}$ be a sequence of finite transition matrices. Let $f$ be a probability vector and denote $f^{(l,j)} = f \cdot P_l \cdots P_{l+j}$. If $\lim_{t \to \infty} P_t = P$, then $\lim_{j \to \infty} f^{(l,j)} = g$ for all $l > 0$.*

Using this lemma and Theorem 1, we can prove the following theorem.

**Theorem 3 (BAP with GLIE)** *On a WAGB G, w.p.1, BAP with GLIE exploration (and $m \geq k(L_G + 2)$) converges to either a strict Nash equilibrium or a Nash equilibrium in D.*

## 4.2 Learning the virtual game

So far, we have shown that if the game structure is known in a WAGB, then BAP will converge to the terminal state. To prove optimal convergence of the OAL algorithm, we need to further demonstrate that 1) every virtual game is a WAGB, and 2) in OAL, the "temporary" virtual game $VG_t$ will converge to the "correct" virtual game $VG^*$ w.p.1. The first of these two issues is handled by the following lemma:

**Lemma 4** *The virtual game VG of any n-player team state game is a weakly acyclic game w.r.t a biased set that contains all the optimal Nash equilibria, and no other joint actions. (By the definition of a virtual game, there are no strict Nash equilibria other than optimal ones.) The length of the shortest best-response path $L_{VG} \leq n$.*

Lemma 4 implies that BAP in a known virtual game with GLIE exploration will converge to an optimal Nash equilibrium. This is because (by Theorem 3) BAP in a WAGB will converge to either a Nash equilibrium in a biased set $D$ or a strict Nash equilibrium, and (by Lemma 4) any virtual game is a WAGB with all such Nash equilibria being optimal.

The following two lemmas are the last link of our proof chain. They show that OAL will cause agents to obtain the correct virtual game almost surely.

**Lemma 5** *In any team Markov game, (part 3 of) OAL assures that as $t \to \infty$, $\max_{s \in S, a \in A}(|Q_t(s,a) - Q^*(s,a)|) \leq C\sqrt{\frac{\log \log N_t}{N_t}}$ for some constant $C > 0$ w.p.1.*

Using Lemma 5, the following lemma is easy to prove.

**Lemma 6** *Consider any team Markov game. Let $\theta_t$ be the event that for all $t' > t$, $VG_{t'} = VG^*$ in the OAL algorithm in a given state. If 1) $B(N_t)$ decreases monotonically to zero ($\lim_{t \to \infty} B(N_t) = 0$), and 2) $\lim_{t \to \infty} \frac{\sqrt{\log \log N_t / N_t}}{B(N_t)} = 0$, then $\lim_{t \to \infty} Pr\{\theta_t\} = 1$.*

Lemma 6 states that if the criterion for including a joint action among the $\varepsilon$-optimal joint actions in OAL is not made strict too quickly (quicker than the iterated logarithm), then agents will identify all optimal joint actions with probability one. In this case, they set up the correct virtual game. It is easy to make OAL satisfy this condition. E.g., any function $B(N_t) = N_t^{\eta - \frac{1}{2}}, 0 < \eta < \frac{1}{2}$, will do.

## 4.3 Main convergence theorem

Now we are ready to prove that OAL converges to an optimal Nash equilibrium in any team Markov game, even when the game structure is unknown. The idea is to show that the OAL agents learn the game structure (VGs) and the optimal coordination policy (over these VGs). OAL tackles these two learning problems simultaneously—specifically, it interleaves BAP (with GLIE exploration) with learning of game structure. However, the convergence proof does not make use of this fact. Instead, the proof proceeds by showing that the VGs are learned first, and coordination second (the learning algorithm does not even itself know when the switch occurs, but it does occur w.p.1).

**Theorem 7 (Optimal convergence)** *In any team Markov game among $n$ agents, if (1) $m \geq k(n+2)$, and (2) $B(N_t)$ satisfies Lemma 6, then the OAL algorithm converges to an optimal Nash equilibrium w.p.1.*

**Proof.** According to [1], a team Markov game can be decomposed into a sequence of state games. The optimal equilibria of these state games form the optimal policy $\pi^*$ for the game. By the definition of GLIE exploration, each state in the finite state space will be visited infinitely often w.p.1. Thus, it is sufficient to only prove that the OAL algorithm will converge to the optimal policy over individual state games w.p.1.

Let $\theta_t$ be the event that $VG_{t'} = VG^*$ at that state for all $t' > t$. Let $\varepsilon_1$ be any positive constant. If Condition (2) of the theorem is satisfied, by Lemma 6 there exists a time $T(\varepsilon_1)$ such that $Pr\{\theta_t\} > 1 - \varepsilon_1$ if $t > T(\varepsilon_1)$.

If $\theta_t$ occurs and Condition (1) of the theorem is satisfied, by Theorem 3, OAL will converge to either a strict Nash equilibrium or a Nash equilibrium in the biased set w.p.1. Furthermore, by Lemma 4, we know that the biased set contains all of the optimal Nash equilibria (and nothing else), and there are no strict Nash equilibria outside the biased set. Therefore, if $\theta_t$ occurs, then OAL converges to an optimal Nash equilibrium w.p.1.[3] Let $\varepsilon_2$ be any positive constant, and let $\xi_\tau$ be the event that the agents play an optimal joint action at a given state for all $t' \geq \tau$. With this notation, we can reword the previous sentence: there exists a time $\Upsilon(\varepsilon_2, t)$ such that if $\tau > \Upsilon(\varepsilon_2, t)$, then $Pr\{\xi_\tau | \theta_t\} > 1 - \varepsilon_2$.

Put together, there exists a time $\Psi(\varepsilon_1, \varepsilon_2)$ such that if $\tau > \Psi(\varepsilon_1, \varepsilon_2)$, then $Pr\{\xi_\tau\} > Pr\{\xi_\tau | \theta_t\} Pr\{\theta_t\} > (1 - \varepsilon_1)(1 - \varepsilon_2) > 1 - \varepsilon_1 - \varepsilon_2$. Because $\varepsilon_1$ and $\varepsilon_2$ are only used in the proof (they are not parameters of the OAL algorithm), we can choose them to be arbitrarily small. Therefore, OAL converges to an optimal Nash equilibrium w.p.1. $\square$

# 5 Conclusions and future research

With multiple Nash equilibria, multiagent RL becomes difficult even when agents do not have conflicting interests. In this paper, we present OAL, the first algorithm that converges to an optimal Nash equilibrium with probability 1 in any team Markov game. In the future work, we consider extending the algorithm to some general-sum Markov games.

### Acknowledgments

Wang is supported by NSF grant IIS-0118767, the DARPA OASIS program, and the PASIS project at CMU. Sandholm is supported by NSF CAREER Award IRI-9703122, and NSF grants IIS-9800994, ITR IIS-0081246, and ITR IIS-0121678.

## Footnotes

[1] Throughout the paper, every Nash equilibrium that we discuss is also a subgame perfect Nash equilibrium. (This refinement of Nash equilibrium was first introduced in [13] for different games).

[2] Akin to how we modeled BAP as a stationary Markov chain above, Young modeled adaptive play (AP) as a stationary Markov chain [18]. There are two differences. First, unlike AP's, BAP's action selection is biased. Second, in Young's model, it is possible to have several absorbing states while in our model, at most one absorbing state exists (for any team game, our model has exactly one absorbing state). This is because we cluster all the absorbing states into one. This allows us to prove our main convergence theorem.

[3]Theorem 3 requires $m \geq k(L_{VG} + 2)$. If Condition (1) of our main theorem is satisfied ($m \geq k(n+2)$), then by Lemma 4, we do have $m \geq k(L_{VG} + 2)$.

## References

[1] C.Boutilier. Planning, learning and coordination in multi-agent decision processes. In *TARK*, 1996.
[2] C.Claus and C.Boutilier. The dynamics of reinforcement learning in cooperative multi-agent systems. In *AAAI*, 1998.
[3] D.Fudenberg and D.K.Levine. *The theory of learning in games*. MIT Press, 1998.
[4] D.L.Isaacson and R.W.Madsen. *Markov chain: theory and applications*. John Wiley and Sons, Inc, 1976.
[5] G.Wei$\beta$. Learning to coordinate actions in multi-agent systems. In *IJCAI*, 1993.
[6] J.Hu and W.P.Wellman. Multiagent reinforcement learning: theoretical framework and an algorithm. In *ICML*, 1998.
[7] M.Kandori, G.J.Mailath, and R.Rob. Learning, mutation, and long run equilibria in games. *Econometrica*, 61(1):29–56, 1993.
[8] M.Littman. Friend-or-Foe Q-learning in general sum game. In *ICML*, 2001.
[9] M.L.Littman. Value-function reinforcement learning in markov games. *J. of Cognitive System Research*, 2:55–66, 2000.
[10] M.L.Purterman. *Markov decision processes-discrete stochastic dynamic programming*. John Wiley, 1994.
[11] M.Tan. Multi-agent reinforcement learning: independent vs. cooperative agents. In *ICML*, 1993.
[12] R.A.Howard. *Dynamic programming and Markov processes*. MIT Press, 1960.
[13] R. Selten. Spieltheoretische behandlung eines oligopolmodells mit nachfrageträgheit. *Zeitschrift für die gesamte Staatswissenschaft*, 12:301–324, 1965.
[14] S. Singh, T.Jaakkola, M.L.Littman, and C.Szepesvari. Convergence results for single-step on-policy reinforcement learning algorithms. *Machine Learning*, 2000.
[15] S.Sen, M.Sekaran, and J. Hale. Learning to coordinate without sharing information. In *AAAI*, 1994.
[16] F. Thusijsman. *Optimality and equilibrium in stochastic games*. Centrum voor Wiskunde en Informatica, 1992.
[17] T.Sandholm and R.Crites. Learning in the iterated prisoner's dilemma. *Biosystems*, 37:147–166, 1995.
[18] H. Young. The evolution of conventions. *Econometrica*, 61(1):57–84, 1993.

